# A Bayesian Framework for Figure-Ground Interpretation

**Vicky Froyen**
*Center for Cognitive Science
Rutgers University, Piscataway, NJ 08854
Laboratory of Experimental Psychology
University of Leuven (K.U. Leuven), Belgium
vicky.froyen@eden.rutgers.edu

**Jacob Feldman**
Center for Cognitive Science
Rutgers University, Piscataway, NJ 08854
jacob@ruccs.rutgers.edu

**Manish Singh**
Center for Cognitive Science
Rutgers University, Piscataway, NJ 08854
manish@ruccs.rutgers.edu

## Abstract

Figure/ground assignment, in which the visual image is divided into nearer (figural) and farther (ground) surfaces, is an essential step in visual processing, but its underlying computational mechanisms are poorly understood. Figural assignment (often referred to as border ownership) can vary along a contour, suggesting a spatially distributed process whereby local and global cues are combined to yield local estimates of border ownership. In this paper we model figure/ground estimation in a Bayesian belief network, attempting to capture the propagation of border ownership across the image as local cues (contour curvature and T-junctions) interact with more global cues to yield a figure/ground assignment. Our network includes as a nonlocal factor *skeletal* (medial axis) structure, under the hypothesis that medial structure "draws" border ownership so that borders are owned by the skeletal hypothesis that best explains them. We also briefly present a psychophysical experiment in which we measured local border ownership along a contour at various distances from an inducing cue (a T-junction). Both the human subjects and the network show similar patterns of performance, converging rapidly to a similar pattern of spatial variation in border ownership along contours.

Figure/ground assignment (further referred to as f/g), in which the visual image is divided into nearer (figural) and farther (ground) surfaces, is an essential step in visual processing. A number of factors are known to affect f/g assignment, including region size [9], convexity [7, 16], and symmetry [1, 7, 11]. Figural assignment (often referred to as *border ownership*, under the assumption that the figural side "owns" the border) is usually studied globally, meaning that entire surfaces and their enclosing boundaries are assumed to receive a globally consistent figural status. But recent psychophysical findings [8] have suggested that border ownership can vary locally along a boundary, even leading to a globally inconsistent figure/ground assignment—broadly consistent with electrophysiological evidence showing local coding for border ownership in area V2 as early as 68 msec after image onset [20]. This suggests a spatially distributed and potentially competitive process of figural assignment [15], in which adjacent surfaces compete to own their common boundary, with figural status propagating across the image as this competition proceeds. But both the principles and computational mechanisms underlying this process are poorly understood.

*V.F. was supported by a Fullbright Honorary fellowship and by the Rutgers NSF IGERT program in Perceptual Science, NSF DGE 0549115, J.F. by NIH R01 EY15888, and M.S. by NSF CCF-0541185*

In this paper we consider how border ownership might propagate over both space and time—that is, across the image as well as over the progression of computation. Following Weiss et al. [18] we adopt a Bayesian belief network architecture, with nodes along boundaries representing estimated border ownership, and connections arranged so that both neighboring nodes and nonlocal integrating nodes combine to influence local estimates of border ownership. Our model is novel in two particular respects: (a) we combine both local and global influences on border ownership in an integrated and principled way; and (b) we include as a nonlocal factor *skeletal* (medial axis) influences on f/g assignment. Skeletal structure has not been previously considered as a factor on border ownership, but its relevance follows from a model [4] in which shapes are conceived of as generated by or "grown" from an internal skeleton, with the consequence that their boundaries are perceptually "owned" by the skeletal side.

We also briey present a psychophysical experiment in which we measured local border ownership along a contour, at several distances from a strong local f/g inducing cue, and at several time delays after the onset of the cue. The results show measurable spatial differences in judged border ownership, with judgments varying with distance from the inducer; but no temporal effect, with essentially asymptotic judgments even after very brief exposures. Both results are consistent with the behavior of the network, which converges quickly to an asymptotic but spatially nonuniform f/g assignment.

## 1   The Model

**The Network.**   For simplicity, we take an edge map as input for the model, assuming that edges and T-junctions have already been detected. From this edge map we then create a Bayesian belief network consisting of four hierarchical levels. At the input level the model receives evidence $E$ from the image, consisting of local contour curvature and T-junctions. The nodes for this level are placed at equidistant locations along the contour. At the first level the model estimates local border ownership. The border ownership, or $B$-nodes at this level are at the same locations as the $E$-nodes, but are connected to their nearest neighbors, and are the parent of the $E$-node at their location. (As a simplifying assumption, such connections are broken at T-junctions in such a way that the occluded contour is disconnected from the occluder.) The highest level has skeletal nodes, $S$, whose positions are defined by the circumcenters of the Delaunay triangulation on all the $E$-nodes, creating a coarse medial axis skeleton [13]. Because of the structure of the Delaunay, each $S$-node is connected to exactly three $E$-nodes from which they receive information about the position and the local tangent of the contour. In the current state of the model the $S$-nodes are "passive", meaning their posteriors are computed before the model is initiated. Between the $S$ nodes and the $B$ nodes are the grouping nodes $G$. They have the same positions as the $S$-nodes and the same Delaunay connections, but to $B$-nodes that have the same image positions as the $E$-nodes. They will integrate information from distant $B$-nodes, applying an interiority cue that is influenced by the local strength of skeletal axes as computed by the $S$-nodes (Fig. 1). Although this is a multiply connected network, we have found that given reasonable parameters the model converges to intuitive posteriors for a variety of shapes (see below).

**Updating.**   Our goal is to compute the posterior $p(B_i|I)$, where $I$ is the whole image. $B_i$ is a binary variable coding for the local direction of border ownership, that is, the side that owns the border. In order for border ownership estimates to be influenced by image structure elsewhere in the image, information has to propagate throughout the network. To achieve this propagation, we use standard equations for node updating [14, 12]. However while to all other connections being directed, connections at the $B$-node level are undirected, causing each node to be child and parent node at the same time. Considering only the $B$-node level, a node $B_i$ is only separated from the rest of the network by its two neighbors. Hence the Markovian property applies, in that $B_i$ only needs to get iterative information from its neighbors to eventually compute $p(B_i|I)$. So considering the whole network, at each iteration $t$, $B_i$ receives information from both its child, $E_i$ and from its parents—that is neigbouring nodes ($B_{i+1}$ and $B_{i-1}$)—as well as all grouping nodes connected to it ($G_j, ..., G_m$). The latter encode for interiority versus exteriority, interiority meaning that the B-node's estimated gural direction points towards the G-node in question, exteriority meaning that it points away. Integrating all this information creates a multidimensional likelihood function: $p(B_i|B_{i-1}, B_{i+1}, G_j, ..., G_m)$. Because of its complexity we choose to approximate it (assuming all nodes are marginally independent of each other when conditioned on $B_i$) by

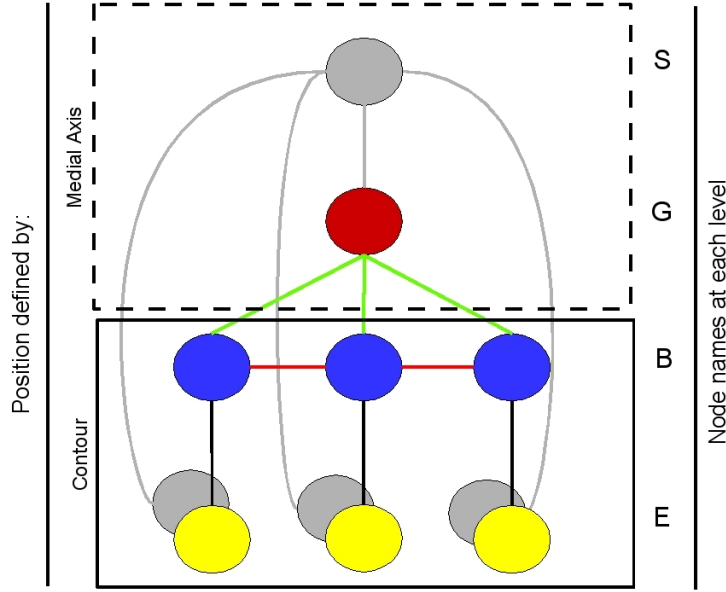

Figure 1: Basic network structure of the model. Both skeletal (*S*-nodes) and border-ownerhsip nodes (*B*-nodes) get evidence from *E*-nodes, though different types. *S*-nodes receive mere positional information, while *B*-nodes receive information about local curvature and the presence of T-junctions. Because of the structure of the Delaunay triangulation *S*-nodes and *G*-nodes (grouping nodes) always get input from exactly three nodes, respectively *E* and *B*-nodes. The gray color depicts the fact that this part of the network is computed before the model is initiated and does not thereafter interact with the dynamics of the model.

$$p(B_i|P_j, ..., P_m) \propto \prod_j^m p(B_i|P_j) \tag{1}$$

where the $P_j$'s are the parents of $B_i$. Given this, at each iteration, each node $B_i$ performs the following computation:

$$Bel(B_i) \leftarrow c\lambda(B_i)\pi(B_i)\alpha(B_i)\beta(B_i) \tag{2}$$

where conceptually $\lambda$ stands for bottom-up information, $\pi$ for top down information and $\alpha$ and $\beta$ for information received from within the same level. More formally,

$$\lambda(B_i) \leftarrow p(E|B_i) \tag{3}$$

$$\pi(B_i) \leftarrow \prod_j^m \sum_{G_j} p(B_i|G_j)\pi_{G_j}(B_i) \tag{4}$$

and analogously to equation 4 for $\alpha(B_i)$ and $\beta(B_i)$, which compute information coming from $B_{i-1}$ and $B_{i+1}$ respectively. For these $\pi_{B_{i-1}}(B_i)$, $\pi_{B_{i+1}}(B_i)$, and $\pi_{G_j}(B_i)$:

$$\pi_{G_j}(B_i) \leftarrow c'\pi(G) \prod_{k \neq i} \lambda_{B_k}(G_j) \tag{5}$$

$$\pi_{B_{i-1}}(B_i) \leftarrow c'\beta(B_{i-1})\lambda(B_{i-1})\pi(B_{i-1}) \tag{6}$$

and $\pi_{B_{i+1}}(B_i)$ is analogous to $\pi_{B_{i-1}}(B_i)$, with $c'$ and $c$ being normalization constants. Finally for the *G*-nodes:

$$Bel(G_i) \leftarrow c\lambda(G_i)\pi(G_i) \tag{7}$$

$$\lambda(G_i) \leftarrow \prod_j \lambda_{B_j}(G_i) \tag{8}$$

$$\lambda_{B_j}(G_i) \leftarrow \sum_{B_j} \lambda(B_j)p(B_i|G_j)[\alpha(B_j)\beta(B_j)\prod_{k \neq i}^{m}\sum_{G_k}p(B_i|G_k)\pi_{G_k}(B_i)] \tag{9}$$

The posteriors of the *S*-nodes are used to compute the $\pi(G_i)$. This posterior computes how well the *S*-node at each position explains the contour—that is, how well it accounts for the cues flowing from the *E*-nodes it is connected to. Each Delaunay connection between *S*- and *E*-nodes can be seen as a rib that sprouts from the skeleton. More specifically each rib sprouts in a direction that is normal (perpendicular) to the tangent of the contour at the *E*-node plus a random error $\phi_i$ chosen independently for each rib from a von Mises distribution centered on zero, i.e. $\phi_i \sim V(0, \kappa_S)$ with spread parameter $\kappa_S$ [4]. The rib lengths are drawn from an exponential decreasing density function $p(\rho_i) \propto e^{-\lambda_S \rho_i}$ [4]. We can now express how well this node "explains" the three *E*-nodes it is connected to via the probability that this *S*-node deserves to be a skeletal node or not,

$$p(S = \text{true}|E_1, E_2, E_3) \propto \prod_i p(\rho_i)p(\phi_i) \tag{10}$$

with $S = \text{true}$ depicting that this *S*-node deserves to be a skeletal node. From this we then compute the prior $\pi(G_i)$ in such a way that good (high posterior) skeletal nodes induce a high interiority bias, hence a stronger tendency to induce figural status. Conversely, bad (low posterior) skeletal nodes create a prior close to indifferent (uniform) and thus have less (or no) influence on figural status.

**Likelihood functions**   Finally we need to express the likelihood function necessary for the updating rules described above. The first two likelihood functions are part of $p(E_i|B_i)$, one for each of the local cues. The first one, reflecting local curvature, gives the probability of the orientations of the two vectors inherent to $E_i$ ($\alpha_1$ and $\alpha_2$) given both direction of figure ($\theta$) encoded in $B_i$ as a von Mises density centered on $\theta$, i.e. $\alpha_i \sim V(\theta, \kappa_{EB})$. The second likelihood function, reflecting the presence of a T-junction, simply assumes a fixed likelihood when a T-junction is present—that is $p(\text{T-junction} = \text{true}|B_i) = \theta_T$, where $B_i$ places the direction of figure in the direction of the occluder. This likelihood function is only in effect when a T-junction is present, replacing the curvature cue at that node.

The third likelihood function serves to keep consistency between nodes of the first level. This function $p(B_i|B_{i-1})$ or $p(B_i|B_{i+1})$ is used to compute $\alpha(B)$ and $\beta(B)$ and is defined 2x2 conditional probability matrix with a single free parameter, $\theta_{BB}$ (the probability that figural direction at both *B*-nodes are the same). A fourth and final likelihood function $p(B_i|G_j)$ serves to propagate information between level one and two. This likelihood function is 2x2 conditional probability matrix matrix with one free parameter, $\theta_{BG}$. In this case $\theta_{BG}$ encodes the probability that the figural direction of the *B*-node is in the direction of the exterior or interior preference of the *G*-node. In total this brings us to six free parameters in the model: $\kappa_S$, $\lambda_S$, $\kappa_{EB}$, $\theta_T$, $\theta_{BB}$, and $\theta_{BG}$.

## 2   Basic Simulations

To evaluate the performance of the model, we first tested it on several basic stimulus configurations in which the desired outcome is intuitively clear: a convex shape, a concave shape, a pair of overlapping shapes, and a pair of non-overlapping shapes (Fig. 2,3). The convex shape is the simplest in that curvature never changes sign. The concave shape includes a region with oppositely signed curvature. (The shape is naturally described as predominantly positively curved with a region of negative curvature, i.e. a concavity. But note that it can also be interpreted as predominantly negatively curved "window" with a region of positive curvature, although this is not the intuitive interpretation.)

The overlapping pair of shapes consists of two convex shapes with one partly occluding the other, creating a competition between the two shapes for the ownership of the common borderline. Finally the non-overlapping shapes comprise two simple convex shapes that do not touch—again setting up a competition for ownership of the two inner boundaries (i.e. between each shape and the ground space between them). Fig. 2 shows the network structures for each of these four cases.

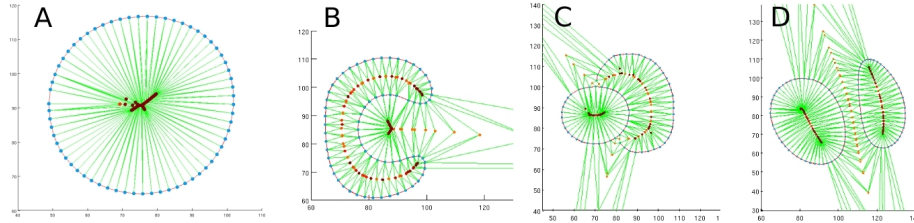

Figure 2: Network structure for the four shape categories (left to right: convex, concave, overlapping, non-overlapping shapes). Blue depict the locations of the *B*-nodes (and also the *E*-nodes), the red connections are the connections between *B*-nodes, the green connections are connections between *B*-nodes and *G*-nodes, and the *G*-nodes (and also the *S*-nodes) go from orange to dark red. This colour code depicts low (orange) to high (dark red) probability that this is a skeletal node, and hence the strength of the interiority cue.

Running our model with hand-estimated parameter values yields highly intuitive posteriors (Fig. 3), an essential "sanity check" to ensure that the network approximates human judgments in simple cases. For the convex shape the model assigns figure to the interior just as one would expect even based solely on local curvature (Fig. 3A). In the concave figure (Fig. 3B), estimated border ownership begins to reverse inside the deep concavity. This may seem surprising, but actually closely matches empirical results obtained when local border ownership is probed psychophysically inside a similarly deep concavity, i.e. a "negative part" in which f/g seems to partly reverse [8]. For the overlapping shapes posteriors were also intuitive, with the occluding shape interpreted as in front and owning the common border (Fig. 3C). Finally, for the two non-overlapping shapes the model computed border-ownership just as one would expect if each shape were run separately, with each shape treated as figural along its entire boundary (Fig. 3D). That is, even though there *is* skeletal structure in the ground-region between the two shapes (see Fig. 2D), its posterior is weak compared to the skeletal structure inside the shapes, which thus loses the competition to own the boundary between them.

For all these configurations, the model not only converged to intuitive estimates but did so rapidly (Fig. 4), always in fewer cycles than would be expected by pure lateral propagation, $n_{\text{iterations}} < N_{\text{nodes}}$ [18] (with these parameters, typically about five times faster).

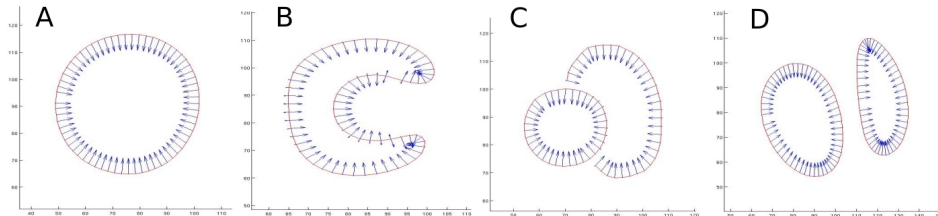

Figure 3: Posteriors after convergence for the four shape categories (left to right: convex, concave, overlapping, non-overlapping). Arrows indicate estimated border ownership, with direction pointing to the perceived figural side, and length proportional to the magnitude of the posterior. All four simulations used the same parameters.

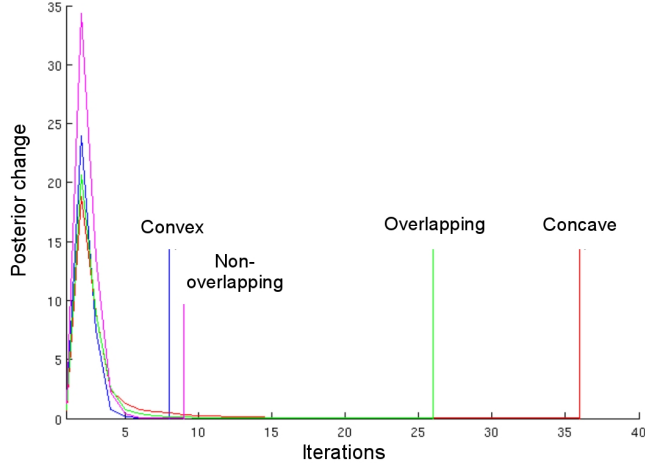

Figure 4: Convergence of the model for the basic shape categories. The vertical lines represent the point of convergence for each of the three shape categories. The posterior change is calculated as $\sum |p(B_i = 1|I)_t - p(B_i = 1|I)_{t-1}|$ at each iteration.

## 3 Comparison to human data

Beyond the simple cases reviewed above, we wished to submit our network to a more fine-grained comparison with human data. To this end we compared its performance to that of human subjects in an experiment we conducted (to be presented in more detail in a future paper). Briefly, our experiment involved finding evidence for propagation of f/g signals across the image. Subjects were first shown a stimulus in which the f/g configuration was globally and locally unambiguous and consistent: a smaller rectangle partly occluding a larger one (Fig. 5A), meaning that the smaller (front) one owns the common border. Then this configuration was perturbed by adding two bars, of which one induced a local f/g reversal—making it now appear locally that the larger rectangle owned the border (Fig. 5B). (The other bar in the display does not alter f/g interpretation, but was included to control for the attentional affects of introducing a bar in the image.) The inducing bar creates T-junctions that serve as strong local f/g cues, in this case tending to reverse the prior global interpretation of the figure. We then measured subjective border ownership along the central contour at various distances from the inducing bar, and at different times after the onset of the bar (25ms, 100ms and 250ms). We measured border ownership locally using a method introduced in [8] in which a local motion probe is introduced at a point on the boundary between two color regions of different colors, and the subject is asked which color appeared to move. Because the figural side "owns" the border, the response reflects perceived figural status.

The goal of the experiment was to actually measure the progression of the influence of the inducing T-junction as it (hypothetically) propagated along the boundary. Briefly, we found no evidence of temporal differences, meaning that f/g judgments were essentially constant over time, suggesting rapid convergence of local f/g assignment. (This is consistent with the very rapid convergence of our network, which would suggest a lack of measurable temporal differences except at much shorter time scales than we measured.) But we did find a progressive reduction of f/g reversal with increasing distance from the inducer—that is, the influence of the T-junction decayed with distance. Mean responses aggregated over subjects (shortest delay only) are shown in Fig. 6.

In order to run our model on this stimulus (which has a much more complex structure than the simple figures tested above) we had to make some adjustments. We removed the bars from the edge map, leaving only the T-junctions as underlying cues. This was a necessary first step because our model is not yet able to cope with skeletons that are split up by occluders. (The larger rectangle's skeleton has been split up by the lower bar.) In this way all contours except those created by the bars were used to create the network (Fig. 7). Given this network we ran the model using hand-picked parameters that

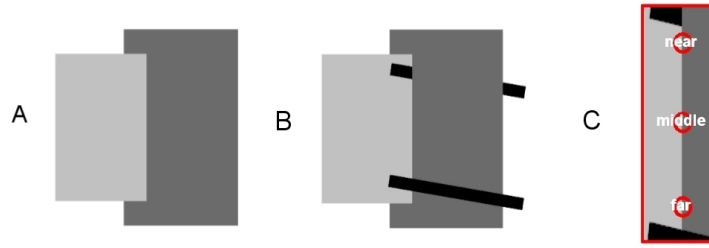

Figure 5: Stimuli used in the experiment. A. Initial stimulus with locally and globally consistent and unambiguous f/g. B. Subsequently bars were added of which one (the top bar in this case) created a local reversal of f/g. C. Positions at which local f/g judgments of subjects were probed.

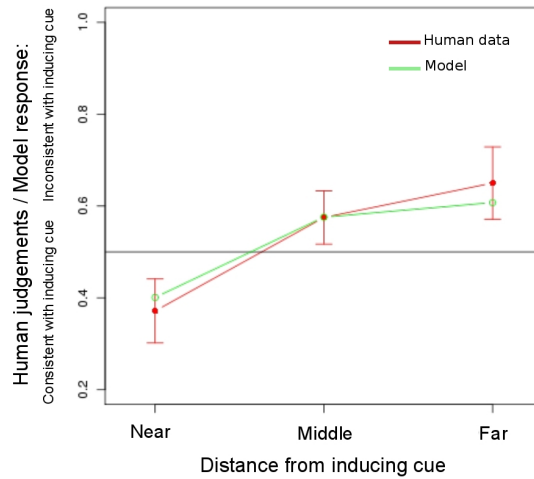

Figure 6: Results from our experiment aggregated for all 7 subjects (shortest delay only) are shown in red. The $x$-axis shows distance from the inducing bar at which f/g judgment was probed. The $y$-axis shows the proportion of trials on which subjects judged the smaller rectangle to own the boundary. As can be seen, the further from the T-junction, the lower the f/g reversal. The fitted model (green curve) shows very similar pattern. Horizontal black line indicates chance performance (ambiguous f/g).

gave us the best possible qualitative similarity to the human data. The parameters used never entailed total elimination of the influence of any likelihood function ($\kappa_S = 16$, $\lambda_S = .025$, $\kappa_{EB} = .5$, $\theta_T = .9$, $\theta_{BB} = .9$, and $\theta_{BG} = .6$). As can be seen in Fig. 6 the border-ownership estimates at the locations where we had data show compelling similarities to human judgments. Furthermore along the entire contour the model converged to intuitive border-ownership estimates (Fig. 7) very rapidly (within 36 iterations). The fact that our model yielded intuitive estimates for the current network in which not all contours were completed shows another strength of our model. Because our model included grouping nodes, it did not require contours to be amodally completed [6] in order for information to propagate.

## 4  Conclusion

In this paper we proposed a model rooted in Bayesian belief networks to compute figure/ground. The model uses both local and global cues, combined in a principled way, to achieve a stable and apparently psychologically reasonable estimate of border ownership. Local cues included local curvature and T-junctions, both well-established cues to f/g. Global cues included skeletal structure,

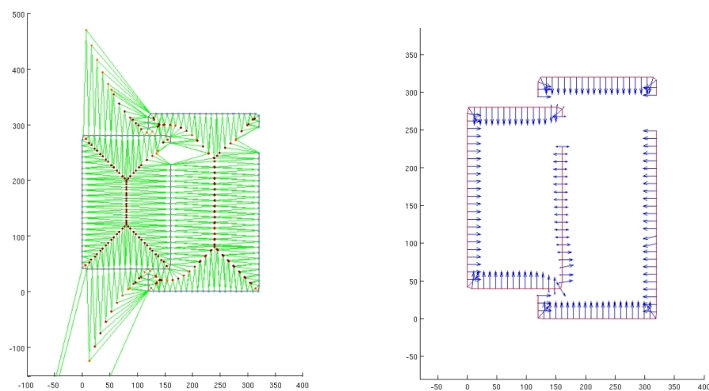

Figure 7: (left) Node structure for the experimental stimulus. (right) The model's local border-ownership estimates after convergence.

a novel cue motivated by the idea that strongly axial shapes tend to be figural and thus own their boundaries. We successfully tested this model on both simple displays, in which it gave intuitive results, and on a more complex experimental stimulus, in which it gave a close match to the pattern of f/g propagation found in our subjects. Specifically, the model, like the human subjects rapidly converged to a stable local f/g interpretation.

Our model's structure shows several interesting parallels to properties of neural coding of border ownership in visual cortex. Some cortical cells (end-stopped cells) appear to code for local curvature [3] and T-junctions [5]. The $B$-nodes in our model could be seen as corresponding to cells that code for border ownership [20]. Furthermore, some authors [2] have suggested that recurrent feedback loops between border ownership cells in V2 and cells in V4 (corresponding to $G$-nodes in our model) play a role in the rapid computation of border ownership. The very rapid convergence we observed in our model likewise appears to be due to the connections between $B$-nodes and $G$-nodes. Finally scale-invariant shape representations (such as, speculatively, those based on skeletons) are thought to be present in higher cortical regions such as IT [17], which project down to earlier areas in ways that are not yet understood.

A number of parallels to past models of f/g should be mentioned. Weiss [18] pioneered the application of belief networks to the f/g problem, though their network only considered a more restricted set of local cues and no global ones, such that information only propagated along the contour. Furthermore it has not been systematically compared to human judgments. Kogo et al. [10] proposed an exponential decay of f/g signals as they spread throughout the image. Our model has a similar decay for information going through the $G$-nodes, though it is also influenced by an angular factor defined by the position of the skeletal node. Like the model by Li Zhaoping [19], our model includes horizontal propagation between $B$-nodes, analogous to border-ownership cells in her model. A neurophysiological model by Craft et al. [2] defines grouping cells coding for an interiority preference that decays with the size of the receptive fields of these grouping cells. Our model takes this a step further by including shape (skeletal) structure as a factor in interiority estimates, rather than simply size of receptive fields (which is similar to the rib lengths in our model).

Currently, our use of skeletons as shape representations is still limited to medial axis skeletons and surfaces that are not split up by occluders. Our future goals including integrating skeletons in a more robust way following the probabilistic account suggested by Feldman and Singh [4]. Eventually, we hope to fully integrate skeleton computation with f/g computation so that the more general problem of shape and surface estimation can be approached in a coherent and unified fashion.

# References

[1] P. Bahnsen. Eine untersuchung uber symmetrie und assymmetrie bei visuellen wahrnehmungen. *Zeitschrift fur psychology*, 108:129–154, 1928.

[2] E. Craft, H. Schütze, E. Niebur, and R. von der Heydt. A neural model of figure-ground organization. *Journal of Neurophysiology*, 97:4310–4326, 2007.

[3] A. Dobbins, S. W. Zucker, and M. S. Cyander. Endstopping and curvature. *Vision Research*, 29:1371–1387, 1989.

[4] J. Feldman and M. Singh. Bayesian estimation of the shape skeleton. *Proceedings of the National Academy of Sciences*, 103:18014–18019, 2006.

[5] B. Heider, V. Meskenaite, and E. Peterhans. Anatomy and physiology of a neural mechanism defining depth order and contrast polarity at illusory contours. *European Journal of Neuroscience*, 12:4117–4130, 2000.

[6] G. Kanizsa. *Organization inVision*. New York: Praeger, 1979.

[7] G. Kanizsa and W. Gerbino. *Vision and Artifact*, chapter Convexity and symmetry in figure-ground organisation, pages 25–32. New York: Springer, 1976.

[8] S. Kim and J. Feldman. Globally inconsistent figure/ground relations induced by a negative part. *Journal of Vision*, 9:1534–7362, 2009.

[9] K. Koffka. *Principles of Gestalt Psychology*. Lund Humphries, London, 1935.

[10] N. Kogo, C. Strecha, L. Van Gool, and J. Wagemans. Surface construction by a 2-d differentiation-integration process: a neurocomputational model for perceived border ownership, depth, and lightness in kanizsa figures. *Psychological Review*, 117:406–439, 2010.

[11] B. Machielsen, M. Pauwels, and J. Wagemans. The role of vertical mirror-symmetry in visual shape detection. *Journal of Vision*, 9:1–11, 2009.

[12] K. Murphy, Y. Weiss, and M.I. Jordan. Loopy belief propagation for approximate inference: an empirical study. *Proceedings of Uncertainty in AI*, pages 467–475, 1999.

[13] R. L. Ogniewicz and O. Kübler. Hierarchic Voronoi skeletons. *Pattern Recognition*, 28:343–359, 1995.

[14] J. Pearl. *Probabilistic reasoning in intelligent systems: networks of plausible inference*. Morgan Kaufmann, 1988.

[15] M. A. Peterson and E. Skow. Inhibitory competition between shape properties in figure-ground perception. *Journal of Experimental Psychology: Human Perception and Performance*, 34:251–267, 2008.

[16] K. A. Stevens and A. Brookes. The concave cusp as a determiner of figure-ground. *Perception*, 17:35–42, 1988.

[17] K. Tanaka, H. Saito, Y. Fukada, and M. Moriya. Coding visual images of object in the inferotemporal cortex of the macaque monkey. *Journal of Neurophysiology*, 66:170–189, 1991.

[18] Y. Weiss. Interpreting images by propagating Bayesian beliefs. *Adv. in Neural Information Processing Systems*, 9:908915, 1997.

[19] L. Zhaoping. Border ownership from intracortical interactions in visual area V2. *Neuron*, 47(1):143–153, Jul 2005.

[20] H. Zhou, H. S. Friedman, and R. von der Heydt. Coding of border ownerschip in monkey visual cortex. *The Journal of Neuroscience*, 20:6594–6611, 2000.

